# Second order derivatives for network pruning: Optimal Brain Surgeon

**Babak Hassibi\*** and **David G. Stork**
Ricoh California Research Center
2882 Sand Hill Road, Suite 115
Menlo Park, CA 94025-7022
stork@crc.ricoh.com

and

\* Department of Electrical Engineering
Stanford University
Stanford, CA 94305

## Abstract

We investigate the use of information from *all* second order derivatives of the error function to perform network pruning (i.e., removing unimportant weights from a trained network) in order to improve generalization, simplify networks, reduce hardware or storage requirements, increase the speed of further training, and in some cases enable rule extraction. Our method, Optimal Brain Surgeon (OBS), is significantly better than magnitude-based methods and Optimal Brain Damage [Le Cun, Denker and Solla, 1990], which often remove the wrong weights. OBS permits the pruning of more weights than other methods (for the same error on the training set), and thus yields better generalization on test data. Crucial to OBS is a recursion relation for calculating the inverse Hessian matrix $H^{-1}$ from training data and structural information of the net. OBS permits a 90%, a 76%, and a 62% reduction in weights over backpropagation with weight decay on three benchmark MONK's problems [Thrun et al., 1991]. Of OBS, Optimal Brain Damage, and magnitude-based methods, only OBS deletes the correct weights from a trained XOR network in every case. Finally, whereas Sejnowski and Rosenberg [1987] used 18,000 weights in their NETtalk network, we used OBS to prune a network to just 1560 weights, yielding better generalization.

## 1  Introduction

A central problem in machine learning and pattern recognition is to minimize the system complexity (description length, VC-dimension, etc.) consistent with the training data. In neural networks this regularization problem is often cast as minimizing the number of connection weights. Without such weight elimination overfitting problems and thus poor generalization will result. Conversely, if there are too few weights, the network might not be able to learn the training data.

If we begin with a trained network having too many weights, the questions then become: Which weights should be eliminated? How should the remaining weights be adjusted for best performance? How can such network pruning be done in a computationally efficient way?

Magnitude based methods [Hertz, Krogh and Palmer, 1991] eliminate weights that have the smallest magnitude. This simple and naively plausible idea unfortunately often leads to the elimination of the wrong weights — small weights can be necessary for low error. Optimal Brain Damage [Le Cun, Denker and Solla, 1990] uses the criterion of minimal increase in training error for weight elimination. For computational simplicity, OBD assumes that the Hessian matrix is diagonal; in fact, however, Hessians for every problem we have considered are strongly *non*-diagonal, and this leads OBD to eliminate the wrong weights. The superiority of the method described here — Optimal Brain Surgeon — lies in great part to the fact that it makes no restrictive assumptions about the form of the network's Hessian, and thereby eliminates the correct weights. Moreover, unlike other methods, OBS does not demand (typically slow) retraining after the pruning of a weight.

## 2 Optimal Brain Surgeon

In deriving our method we begin, as do Le Cun, Denker and Solla [1990], by considering a network trained to a local minimum in error. The functional Taylor series of the error with respect to weights (or parameters, see below) is:

$$\delta E = \left(\frac{\partial E}{\partial \mathbf{w}}\right)^T \cdot \delta \mathbf{w} + \frac{1}{2} \delta \mathbf{w}^T \cdot \mathbf{H} \cdot \delta \mathbf{w} + O(\|\delta \mathbf{w}\|^3) \tag{1}$$

where $\mathbf{H} \equiv \partial^2 E / \partial \mathbf{w}^2$ is the Hessian matrix (containing all second order derivatives) and the superscript T denotes vector transpose. For a network trained to a local minimum in error, the first (linear) term vanishes; we also ignore the third and all higher order terms. Our goal is then to set one of the weights to zero (which we call $w_q$) to minimize the increase in error given by Eq. 1. Eliminating $w_q$ is expressed as:

$$\delta w_q + w_q = 0 \qquad \text{or more generally} \quad \mathbf{e}_q^T \cdot \delta \mathbf{w} + w_q = 0 \tag{2}$$

where $\mathbf{e}_q$ is the unit vector in weight space corresponding to (scalar) weight $w_q$. Our goal is then to solve:

$$Min_q \{ Min_{\delta \mathbf{w}} \{ \tfrac{1}{2} \delta \mathbf{w}^T \cdot \mathbf{H} \cdot \delta \mathbf{w} \} \quad \text{such that} \quad \mathbf{e}_q^T \cdot \delta \mathbf{w} + w_q = 0 \} \tag{3}$$

To solve Eq. 3 we form a Lagrangian from Eqs. 1 and 2:

$$L = \tfrac{1}{2} \delta \mathbf{w}^T \cdot \mathbf{H} \cdot \delta \mathbf{w} + \lambda (\mathbf{e}_q^T \cdot \delta \mathbf{w} + w_q) \tag{4}$$

where $\lambda$ is a Lagrange undetermined multiplier. We take functional derivatives, employ the constraints of Eq. 2, and use matrix inversion to find that the optimal weight change and resulting change in error are:

$$\delta \mathbf{w} = -\frac{w_q}{[\mathbf{H}^{-1}]_{qq}} \mathbf{H}^{-1} \cdot \mathbf{e}_q \qquad and \qquad L_q = \frac{1}{2} \frac{w_q^2}{[\mathbf{H}^{-1}]_{qq}} \tag{5}$$

Note that neither $\mathbf{H}$ nor $\mathbf{H}^{-1}$ need be diagonal (as is assumed by Le Cun et al.); moreover, our method recalculates the magnitude of *all* the weights in the network, by the left side of Eq. 5. We call $L_q$ the "saliency" of weight q — the increase in error that results when the weight is eliminated — a definition more general than Le Cun et al.'s, and which includes theirs in the special case of diagonal $\mathbf{H}$.

Thus we have the following algorithm:

### Optimal Brain Surgeon procedure

1. Train a "reasonably large" network to minimum error.
2. Compute $\mathbf{H}^{-1}$.
3. Find the q that gives the smallest saliency $L_q = w_q^2/(2[\mathbf{H}^{-1}]_{qq})$. If this candidate error increase is much smaller than $E$, then the q[th] weight should be deleted, and we proceed to step 4; otherwise go to step 5. (Other stopping criteria can be used too.)
4. Use the q from step 3 to update *all* weights (Eq. 5). Go to step 2.
5. No more weights can be deleted without large increase in E. (At this point it may be desirable to retrain the network.)

Figure 1 illustrates the basic idea. The relative magnitudes of the error after pruning (before retraining, if any) depend upon the particular problem, but to second order obey: E(mag) ≥ E(OBD) ≥ E(OBS), which is the key to the superiority of OBS. In this example OBS and OBD lead to the elimination of the same weight (weight 1). In many cases, however, OBS will eliminate *different* weights than those eliminated by OBD (cf. Sect. 6). We call our method Optimal Brain *Surgeon* because in addition to deleting weights, it

calculates and *changes* the strengths of other weights without the need for gradient descent or other incremental retraining.

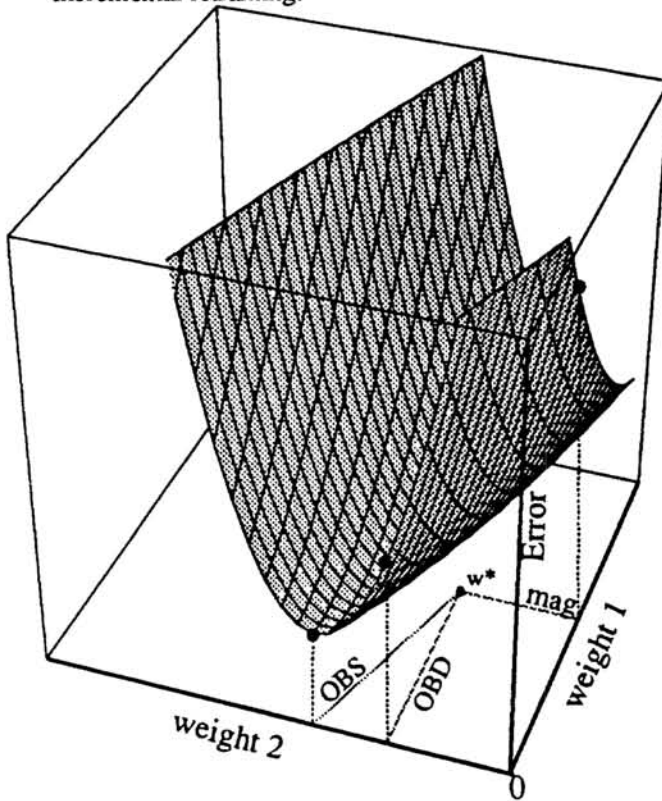

**Figure 1**: Error as a function of two weights in a network. The (local) minimum occurs at weight w*, found by gradient descent or other learning method. In this illustration, a magnitude based pruning technique (mag) then removes the smallest weight, weight 2; Optimal Brain Damage before retraining (OBD) removes weight 1. In contrast, our Optimal Brain Surgeon method (OBS) not only removes weight 1, but *also* automatically adjusts the value of weight 2 to minimize the error, without retraining. The error surface here is general in that it has different curvatures (second derivatives) along different directions, a minimum at a non-special weight value, and a non-diagonal Hessian (i.e., principal axes are *not* parallel to the weight axes). We have found (to our surprise) that every problem we have investigated has strongly *non*-diagonal Hessians — thereby explaining the improvment of our method over that of Le Cun et al.

## 3  Computing the inverse Hessian

The difficulty appears to be step 2 in the OBS procedure, since inverting a matrix of thousands or millions of terms seems computationally intractable. In what follows we shall give a general derivation of the inverse Hessian for a *fully trained* neural network. It makes no difference whether it was trained by backpropagation, competitive learning, the Boltzmann algorithm, or any other method, so long as derivatives can be taken (see below). We shall show that the Hessian can be reduced to the sample covariance matrix associated with certain gradient vectors. Furthermore, the gradient vectors necessary for OBS are normally available at small computational cost; the covariance form of the Hessian yields a recursive formula for computing the inverse.

Consider a general non-linear neural network that maps an input vector **in** of dimension $n_i$ into an output vector **o** of dimension $n_o$, according to the following:

$$\mathbf{o} = \mathbf{F}(\mathbf{w}, \mathbf{in}) \tag{6}$$

where **w** is an n dimensional vector representing the neural network's weights or other parameters. We shall refer to **w** as a weight vector below for simplicity and definiteness, but it must be stressed that **w** could represent *any* continuous parameters, such as those describing neural transfer function, weight sharing, and so on. The mean square error corresponding to the training set is defined as:

$$E = \frac{1}{2P} \sum_{k=1}^{P} (\mathbf{t}^{[k]} - \mathbf{o}^{[k]})^T (\mathbf{t}^{[k]} - \mathbf{o}^{[k]}) \tag{7}$$

where P is the number of training patterns, and $\mathbf{t}^{[k]}$ and $\mathbf{o}^{[k]}$ are the desired response and network response for the $k^{th}$ training pattern. The first derivative with respect to **w** is:

$$\frac{\partial E}{\partial \mathbf{w}} = -\frac{1}{P} \sum_{k=1}^{P} \frac{\partial \mathbf{F}(\mathbf{w}, \mathbf{in}^{[k]})}{\partial \mathbf{w}} (\mathbf{t}^{[k]} - \mathbf{o}^{[k]}) \tag{8}$$

and the second derivative or Hessian is:

$$\mathbf{H} \equiv \frac{\partial^2 E}{\partial \mathbf{w}^2} = \frac{1}{P} \sum_{k=1}^{P} \left[ \frac{\partial \mathbf{F}(\mathbf{w}, \mathbf{in}^{[k]})}{\partial \mathbf{w}} \cdot \frac{\partial \mathbf{F}(\mathbf{w}, \mathbf{in}^{[k]})}{\partial \mathbf{w}}^T - \frac{\partial^2 \mathbf{F}(\mathbf{w}, \mathbf{in}^{[k]})}{\partial \mathbf{w}^2} \cdot (\mathbf{t}^{[k]} - \mathbf{o}^{[k]}) \right] \tag{9}$$

Next we consider a network fully trained to a local minimum in error at $w^*$. Under this condition the network response $o^{[k]}$ will be close to the desired response $t^{[k]}$, and hence we neglect the term involving $(t^{[k]} - o^{[k]})$. Even late in pruning, when this error is not small for a single pattern, this approximation can be justified (see next Section). This simplification yields:

$$H = \frac{1}{P} \sum_{k=1}^{P} \frac{\partial F(w, in^{[k]})}{\partial w} \cdot \frac{\partial F(w, in^{[k]})}{\partial w}^{T} \tag{10}$$

If out network has just a single output, we may define the n-dimensional data vector $X^{[k]}$ of derivatives as:

$$X^{[k]} \equiv \frac{\partial F(w, in^{[k]})}{\partial w} \tag{11}$$

Thus Eq. 10 can be written as:    $$H = \frac{1}{P} \sum_{k=1}^{P} X^{[k]} \cdot X^{[k]T} \tag{12}$$

If instead our network has *multiple* output units, then X will be an $n \times n_o$ matrix of the form:

$$X^{[k]} = \frac{\partial F(w, in^{[k]})}{\partial w} = (\frac{\partial F_1(w, in^{[k]})}{\partial w}, ..., \frac{\partial F_{n_o}(w, in^{[k]})}{\partial w}) \quad = (X_1^{[k]}, ..., X_{n_o}^{[k]}) \tag{13}$$

where $F_i$ is the $i^{th}$ component of F. Hence in this multiple output unit case Eq. 10 generalizes to:

$$H = \frac{1}{P} \sum_{k=1}^{P} \sum_{l=1}^{n_o} X_l^{[k]} \cdot X_l^{[k]T} \tag{14}$$

Equations 12 and 14 show that H is the sample covariance matrix associated with the gradient variable X. Equation 12 also shows that for the single output case we can calculate the full Hessian by sequentially adding in successive "component" Hessians as:

$$H_{m+1} = H_m + \frac{1}{P} X^{[m+1]} \cdot X^{[m+1]T} \quad \text{with} \quad H_0 = \alpha I \text{ and } H_P = H \tag{15}$$

But Optimal Brain Surgeon requires the *inverse* of H (Eq. 5). This inverse can be calculated using a standard matrix inversion formula [Kailath, 1980]:

$$(A + B \cdot C \cdot D)^{-1} = A^{-1} - A^{-1} \cdot B \cdot (C^{-1} + D.A^{-1} \cdot B)^{-1} \cdot D \cdot A^{-1} \tag{16}$$

applied to each term in the analogous sequence in Eq. 16:

$$H_{m+1}^{-1} = H_m^{-1} - \frac{H_m^{-1} \cdot X^{[m+1]} \cdot X^{[m+1]T} \cdot H_m^{-1}}{P + X^{[m+1]T} \cdot H_m^{-1} \cdot X^{[m+1]}} \quad \text{with} \quad H_0^{-1} = \alpha^{-1} I \text{ and } H_P^{-1} = H^{-1} \tag{17}$$

and $\alpha$ ($10^{-8} \leq \alpha \leq 10^{-4}$) a small constant needed to make $H_0^{-1}$ meaningful, and to which our method is insensitive [Hassibi, Stork and Wolff, 1993b]. Actually, Eq. 17 leads to the calculation of the inverse of $(H + \alpha I)$, and this corresponds to the introduction of a penalty term $\alpha \|\delta w\|^2$ in Eq. 4. This has the benefit of penalizing large candidate jumps in weight space, and thus helping to insure that the neglecting of higher order terms in Eq. 1 is valid.

Equation 17 permits the calculation of $H^{-1}$ using a *single* sequential pass through the training data $1 \leq m \leq P$. It is also straightforward to generalize Eq. 18 to the multiple output case of Eq. 15: in this case Eq. 15 will have recursions on both the indices m and $l$ giving:

$$H_{m \, l+1} = H_{ml} + \frac{1}{P} X_{l+1}^{[m]} \cdot X_{l+1}^{[m]T}$$

$$H_{m+1 1} = H_{m n_o} + \frac{1}{P} X_1^{[m+1]} \cdot X_1^{[m+1]T} \tag{18}$$

To sequentially calculate $H^{-1}$ for the multiple output case, we use Eq. 16, as before.

## 4  The (t - o) → 0 approximation

The approximation used for Eq. 10 can be justified on computational and functional grounds, even late in pruning when the training error is not negligible. From the computational view, we note first that normally H is degenerate — especially before significant pruning has been done — and its inverse not well defined.

The approximation guarantees that there are no singularities in the calculation of $\mathbf{H}^{-1}$. It also keeps the computational complexity of calculating $\mathbf{H}^{-1}$ the same as that for calculating $\mathbf{H}$ — $O(P n^2)$. In Statistics the approximation is the basis of Fisher's method of scoring and its goal is to replace the true Hessian with its expected value and guarantee that $\mathbf{H}$ is positive definite (thereby avoiding stability problems that can plague Gauss-Newton methods) [Seber and Wild, 1989].

Equally important are the functional justifications of the approximation. Consider a high capactiy network trained to small training error. We can consider the network structure as involving both signal and noise. As we prune, we hope to eliminate those weights that lead to "overfitting," i.e., learning the noise. If our pruning method did *not* employ the $(\mathbf{t} - \mathbf{o}) \rightarrow 0$ approximation, every pruning step (Eqs. 9 and 5) would inject the noise back into the system, by penalizing for noise terms. A different way to think of the approximation is the following. After some pruning by OBS we have reached a new weight vector that is a local minimum of the error (cf. Fig. 1). Even if this error is not negligible, we want to stay as close to *that* value of the error as we can. Thus we imagine a new, effective teaching signal $\mathbf{t}^*$, that would keep the network near this new error minimum. It is then $(\mathbf{t}^* - \mathbf{o})$ that we in effect set to zero when using Eq. 10 instead of Eq. 9.

## 5  OBS and backpropagation

Using the standard terminology from backpropagation [Rumelhart, Hinton and Williams, 1986] and the single output network of Fig. 2, it is straightforward to show from Eq. 11 that the derivative vectors are:

$$\mathbf{X}^{[k]} = \begin{pmatrix} \mathbf{X}_v^{[k]} \\ \mathbf{X}_u^{[k]} \end{pmatrix} \qquad (19)$$

where
$$[\mathbf{X}_v^{[k]}]^T = \left( f'(net^{[k]})o_{j=1}^{[k]}, ..., f'(net^{[k]})o_{n_j}^{[k]} \right) \qquad (20)$$

refers to derivatives with respect to hidden-to-output weights $v_j$ and

$$[\mathbf{X}_u^{[k]}]^T = \left( f'(net^{[k]})f'(net_1^{[k]})v_1^{[k]}o_{i=1}^{[k]}, ..., f'(net^{[k]})f'(net_1^{[k]})v_1^{[k]}o_{n_i}^{[k]}, ..., \right.$$
$$\left. f'(net^{[k]})f'(net_{n_j}^{[k]})v_{n_j}^{[k]}o_1^{[k]}, ..., f'(net^{[k]})f'(net_{n_j}^{[k]})v_{n_j}^{[k]}o_{n_i}^{[k]} \right) \qquad (21)$$

refers to derivatives with respect to input-to-hidden weights $u_{ji}$, and where lexicographical ordering has been used. The neuron nonlinearity is $f(\cdot)$.

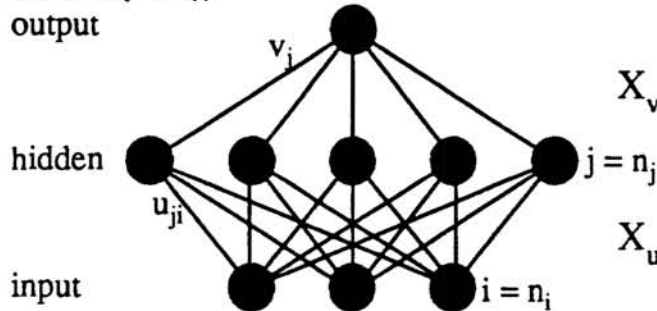

Figure 2: Backpropagation net with $n_i$ inputs and $n_j$ hidden units. The input-to-hidden weights are $u_{ji}$ and hidden-to-output weights $v_j$. The derivative ("data") vectors are $\mathbf{X}_v$ and $\mathbf{X}_u$ (Eqs. 20 and 21).

## 6  Simulation results

We applied OBS, Optimal Brain Damage, and a magnitude based pruning method to the 2-2-1 network with bias unit of Fig. 3, trained on all patterns of the XOR problem. The network was first trained to a local minimum, which had zero error, and then the three methods were used to prune one weight. As shown, the methods deleted different weights. We then trained the original XOR network from different initial conditions, thereby leading to a different local minima. Whereas there were some cases in which OBD or magnitude methods deleted the correct weight, only OBS deleted the correct weight in *every* case. Moreover, OBS changed the values of the remaining weights (Eq. 5) to achieve perfect performance *without any retraining by the backpropagation algorithm*. Figure 4 shows the Hessian of the trained but unpruned XOR network.

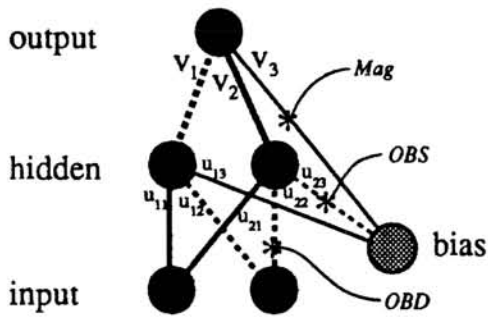

output

hidden

input

bias

**Figure 3**: A nine weight XOR network trained to a local minimum. The thickness of the lines indicates the weight magnitudes, and inhibitory weights are shown dashed. Subsequent pruning using a magnitude based method (*Mag*) would delete weight $v_3$; using Optimal Brain Damage (*OBD*) would delete $u_{22}$. Even with retraining, the network pruned by those methods cannot learn the XOR problem. In contrast, Optimal Brain Surgeon (*OBS*) deletes $u_{23}$ and furthermore changed all other weights (cf. Eq. 5) to achieve zero error on the XOR problem.

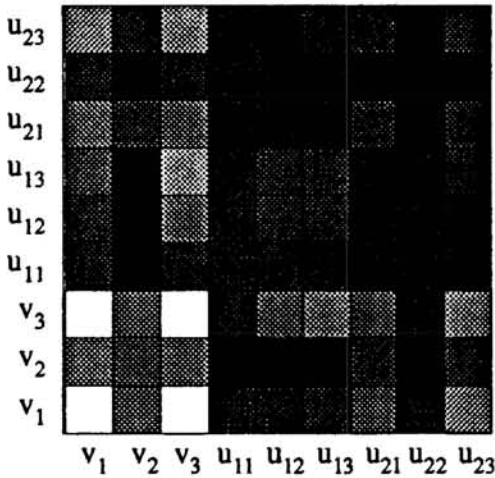

$u_{23}$
$u_{22}$
$u_{21}$
$u_{13}$
$u_{12}$
$u_{11}$
$v_3$
$v_2$
$v_1$

$v_1$ $v_2$ $v_3$ $u_{11}$ $u_{12}$ $u_{13}$ $u_{21}$ $u_{22}$ $u_{23}$

**Figure 4**: The Hessian of the trained but unpruned XOR network, calculated by means of Eq. 12. White represents large values and black small magnitudes. The rows and columns are labeled by the weights shown in Fig. 3. As is to be expected, the hidden-to-output weights have significant Hessian components. Note especially that the Hessian is far from being diagonal. The Hessians for all problems we have investigated, including the MONK's problems (below), are far from being diagonal.

Figure 5 shows two-dimensional "slices" of the nine-dimensional error surface in the neighborhood of a local minimum at w* for the XOR network. The cuts compare the weight elimination of Magnitude methods (left) and OBD (right) with the elimination and weight adjustment given by OBS.

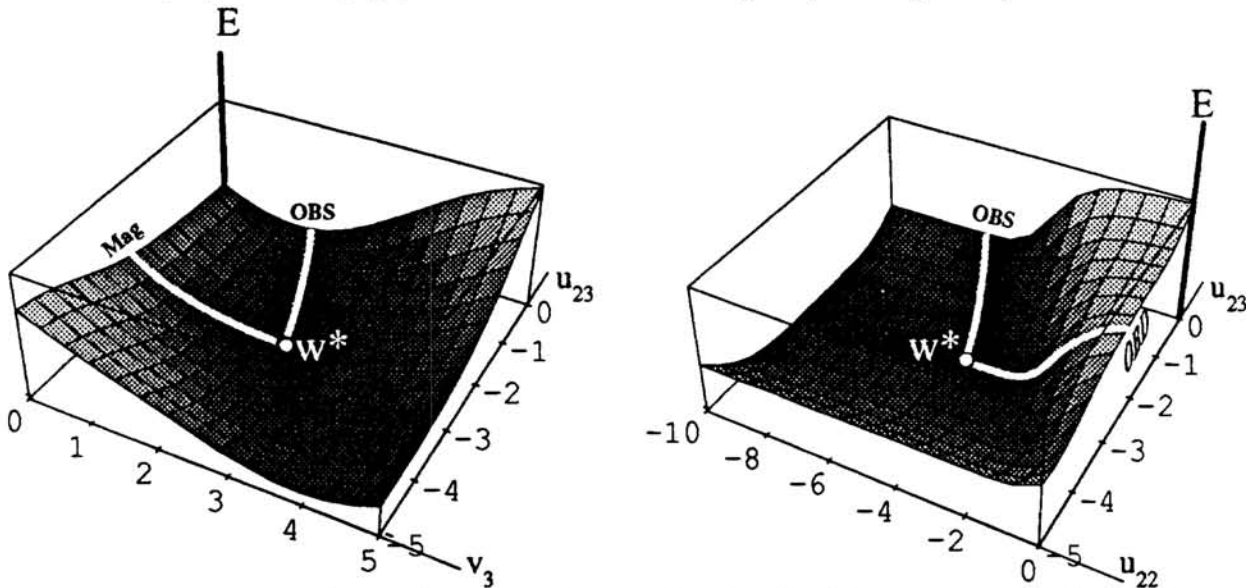

**Figure 5**: (Left) the XOR error surface as a function of weights $v_3$ and $u_{23}$ (cf. Fig. 4). A magnitude based pruning method would delete weight $v_3$ whereas OBS deletes $u_{23}$. (Right) The XOR error surface as a function of weights $u_{22}$ and $u_{23}$. Optimal Brain Damage would delete $u_{22}$ whereas OBS deletes $u_{23}$. For this minimum, only deleting $u_{23}$ will allow the pruned network to solve the XOR problem.

After all network weights are updated by Eq. 5 the system is at zero error (not shown). It is especially noteworthy that in neither case of pruning by magnitude methods nor Optimal Brain Damage will further retraining by gradient descent reduce the training error to zero. In short, magnitude methods and Optimal Brain Damage delete the wrong weights, and their mistake cannot be overcome by further network training. Only Optimal Brain Surgeon deletes the correct weight.

We also applied OBS to larger problems, three MONK's problems, and compared our results to those of Thrun et al. [1991], whose backpropagation network outperformed all other approaches (network and rule-based) on these benchmark problems in an extensive machine learning competition.

| | | Accuracy | | # weights |
| | | training | testing | |
|---|---|---|---|---|
| MONK 1 | BPWD | 100 | 100 | 58 |
| | OBS | 100 | 100 | 14 |
| MONK 2 | BPWD | 100 | 100 | 39 |
| | OBS | 100 | 100 | 15 |
| MONK 3 | BPWD | 93.4 | 97.2 | 39 |
| | OBS | 93.4 | 97.2 | 4 |

Table 1: The accuracy and number of weights found by backpropagation with weight decay (BPWD) found by Thrun et al. [1991], and by OBS on three MONK's problems.

Table 1 shows that for the same performance, OBS (without retraining) required only 24%, 38% and 10% of the weights of the backpropagation network, which was already regularized with weight decay (Fig. 6). The error increase $L$ (Eq. 5) accompanying pruning by OBS negligibly affected accuracy.

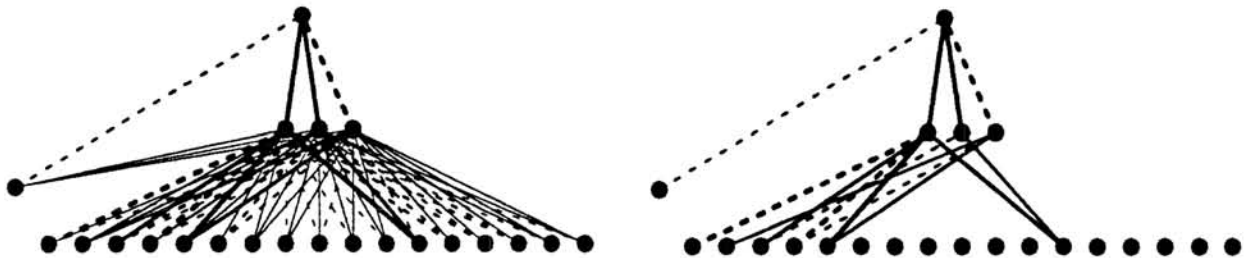

Figure 6: Optimal networks found by Thrun using backpropagation with weight decay (Left) and by OBS (Right) on MONK 1, which is based on logical rules. Solid (dashed) lines denote excitatory (inhibitory) connections; bias units are at left.

The dramatic reduction in weights achieved by OBS yields a network that is simple enough that the logical rules that generated the data can be recovered from the pruned network, for instance by the methods of Towell and Shavlik [1992]. Hence OBS may help to address a criticism often levied at neural networks: the fact that they may be unintelligible.

We applied OBS to a three-layer NETtalk network. While Sejnowski and Rosenberg [1987] used 18,000 weights, we began with just 5546 weights, which after backpropagation training had a test error of 5259. After pruning this net with OBS to 2438 weights, and then retraining and pruning again, we achieved a net with only 1560 weights and test error of only 4701 — a significant improvement over the original, more complex network [Hassibi, Stork and Wolff, 1993a]. Thus OBS can be applied to real-world pattern recognition problems such as speech recognition and optical character recognition, which typically have several thousand parameters.

## 7    Analysis and conclusions

Why is Optimal Brain Surgeon so successful at reducing excess degrees of freedom? Conversely, given this new standard in weight elimination, we can ask: Why are magnitude based methods so poor? Consider again Fig. 1. Starting from the local minimum at w*, a magnitude based method deletes the wrong weight, weight 2, and through retraining, weight 1 will *increase*. The final "solution" is weight 1 → large, weight 2 = 0. This is precisely the *opposite* of the solution found by OBS: weight 1 = 0, weight 2 → large. Although the actual difference in error shown in Fig. 1 may be small, in large networks, differences from many incorrect weight elimination decisions can add up to a significant increase in error.

But most importantly, it is simply wishful thinking to believe that after the elimination of many incorrect weights by magnitude methods the net can "sort it all out" through further training and reach a global optimum, especially if the network has already been pruned significantly (cf. XOR discussion, above).

We have also seen how the approximation employed by Optimal Brain Damage — that the diagonals of the Hessian are dominant — does not hold for the problems we have investigated. There are typically many off-diagonal terms that are comparable to their diagonal counterparts. This explains why OBD often deletes the wrong weight, while OBS deletes the correct one.

We note too that our method is quite general, and subsumes previous methods for weight elimination. In our terminology, magnitude based methods assume isotropic Hessian ($\mathbf{H} \propto \mathbf{I}$); OBD assumes diagonal $\mathbf{H}$; FARM [Kung and Hu, 1991] assumes linear f(net) and only updates the hidden-to-output weights. We have shown that none of those assumptions are valid nor sufficient for optimal weight elimination.

We should also point out that our method is even more general than presented here [Hassibi, Stork and Wolff, 1993b]. For instance, rather than pruning a weight (parameter) by setting it to zero, one can instead reduce a degree of freedom by projecting onto an *arbitrary* plane, e.g., $w_q$ = a constant, though such networks typically have a large description length [Rissanen, 1978]. The pruning constraint $w_q = 0$ discussed throughout this paper makes retraining (if desired) particularly simple. *Several* weights can be deleted simultaneously; bias weights can be exempt from pruning, and so forth. A slight generalization of OBS employs cross-entropy or the Kullback-Leibler error measure, leading to Fisher Information matrix rather than the Hessian (Hassibi, Stork and Wolff, 1993b). We note too that OBS does not by itself give a criterion for when to stop pruning, and thus OBS can be utilized with a wide variety of such criteria. Moreover, gradual methods such as weight decay during learning can be used in conjunction with OBS.

## Acknowledgements
The first author was supported in part by grants AFOSR 91-0060 and DAAL03-91-C-0010 to T. Kailath, who in turn provided constant encouragement. Deep thanks go to Greg Wolff (Ricoh) for assistance with simulations and analysis, and Jerome Friedman (Stanford) for pointers to relevant statistics literature.

## REFERENCES

Hassibi, B. Stork, D. G. and Wolff, G. (1993a). Optimal Brain Surgeon and general network pruning (submitted to ICNN, San Francisco)

Hassibi, B. Stork, D. G. and Wolff, G. (1993b). Optimal Brain Surgeon, Information Theory and network capacity control (in preparation)

Hertz, J., Krogh, A. and Palmer, R. G. (1991). *Introduction to the Theory of Neural Computation* Addison-Wesley.

Kailath, T. (1980). *Linear Systems* Prentice-Hall.

Kung, S. Y. and Hu, Y. H. (1991). A Frobenius approximation reduction method (FARM) for determining the optimal number of hidden units, *Proceedings of the IJCNN-91* Seattle, Washington.

Le Cun, Y., Denker, J. S. and Solla, S. A. (1990). Optimal Brain Damage, in *Proceedings of the Neural Information Processing Systems-2*, D. S. Touretzky (ed.) 598-605, Morgan-Kaufmann.

Rissanen, J. (1978). Modelling by shortest data description, *Automatica* 14, 465-471.

Rumelhart, D. E., Hinton, G. E., and Williams, R. J. (1986). Learning Internal representations by error propagation, Chapter 8 (318-362) in *Parallel Distributed Processing I* D. E. Rumelhart and J. L. McClelland (eds.) MIT Press.

Seber, G. A. F. and Wild, C. J. (1989). *Nonlinear Regression* 35-36 Wiley.

Sejnowski, T. J., and Rosenberg, C. R. (1987). Parallel networks that learn to pronounce English text, *Complex Systems* 1, 145-168.

Thrun, S. B. and 23 co-authors (1991). The MONK's Problems — A performance comparison of different learning algorithms, CMU-CS-91-197 Carnegie-Mellon U. Department of Computer ScienceTech Report.

Towell, G. and Shavlik, J. W. (1992). Interpretation of artificial neural networks: Mapping knowledge-based neural networks into rules, in *Proceedings of the Neural Information Processing Systems-4*, J. E. Moody, D. S. Touretzky and R. P. Lippmann (eds.) 977-984, Morgan-Kaufmann.
